# Learning with Recursive Perceptual Representations

**Oriol Vinyals**
UC Berkeley
Berkeley, CA

**Yangqing Jia**
UC Berkeley
Berkeley, CA

**Li Deng**
Microsoft Research
Redmond, WA

**Trevor Darrell**
UC Berkeley
Berkeley, CA

## Abstract

Linear Support Vector Machines (SVMs) have become very popular in vision as part of state-of-the-art object recognition and other classification tasks but require high dimensional feature spaces for good performance. Deep learning methods can find more compact representations but current methods employ multilayer perceptrons that require solving a difficult, non-convex optimization problem. We propose a deep non-linear classifier whose layers are SVMs and which incorporates random projection as its core stacking element. Our method learns layers of linear SVMs recursively transforming the original data manifold through a random projection of the weak prediction computed from each layer. Our method scales as linear SVMs, does not rely on any kernel computations or nonconvex optimization, and exhibits better generalization ability than kernel-based SVMs. This is especially true when the number of training samples is smaller than the dimensionality of data, a common scenario in many real-world applications. The use of random projections is key to our method, as we show in the experiments section, in which we observe a consistent improvement over previous –often more complicated– methods on several vision and speech benchmarks.

## 1   Introduction

In this paper, we focus on the learning of a general-purpose non-linear classifier applied to perceptual signals such as vision and speech. The Support Vector Machine (SVM) has been a popular method for multimodal classification tasks since its introduction, and one of its main advantages is the simplicity of training a linear model. Linear SVMs often fail to solve complex problems however, and with non-linear kernels, SVMs usually suffer from speed and memory issues when faced with very large-scale data, although techniques such as non-convex optimization [6] or spline approximations [19] exist for speed-ups. In addition, finding the "oracle" kernel for a specific task remains an open problem, especially in applications such as vision and speech.

Our aim is to design a classifier that combines the simplicity of the linear Support Vector Machine (SVM) with the power derived from deep architectures. The new technique we propose follows the philosophy of "stacked generalization" [23], i.e. the framework of building layer-by-layer architectures, and is motivated by the recent success of a convex stacking architecture which uses a simplified form of neural network with closed-form, convex learning [10]. Specifically, we propose a new stacking technique for building a deep architecture, using a linear SVM as the base building block, and a random projection as its core stacking element.

The proposed model, which we call the Random Recursive SVM ($R^2SVM$), involves an efficient, feed-forward convex learning procedure. The key element in our convex learning of each layer is to randomly project the predictions of the previous layer SVM back to the original feature space. As we will show in the paper, this could be seen as recursively transforming the original data manifold so that data from different classes are moved apart, leading to better linear separability in the subsequent layers. In particular, we show that randomly generating projection parameters, instead of fine-tuning them using backpropagation, suffices to achieve a significant performance gain. As a result, our

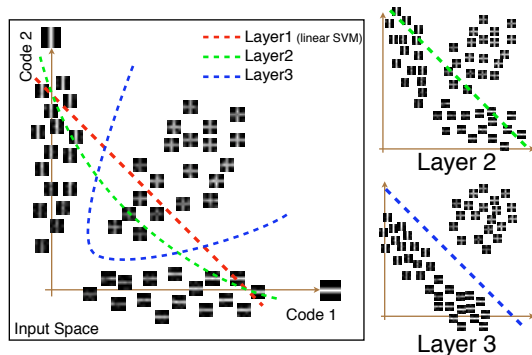

Figure 1: A conceptual example of Random Recursive SVM separating edges from cross-bars. Starting from data manifolds that are not linearly separable, our method transforms the data manifolds in a stacked way to find a linear separating hyperplane in the high layers, which corresponds to non-linear separating hyperplanes in the lower layers. Non-linear classification is achieved without kernelization, using a recursive architecture.

model does not require any complex learning techniques other than training linear SVMs, while canonical deep architectures usually require carefully designed pre-training and fine-tuning steps, which often depend on specific applications.

Using linear SVMs as building blocks our model scales in the same way as the linear SVM does, enabling fast computation during both training and testing time. While linear SVM fails to solve non-linearly separable problems, the simple non-linearity in our algorithm, introduced with sigmoid functions, is shown to adapt to a wide range of real-world data with the same learning structure. From a kernel based perspective, our method could be viewed as a special non-linear SVM, with the benefit that the non-linear kernel naturally emerges from the stacked structure instead of being defined as in conventional algorithms. This brings additional flexibility to the applications, as task-dependent kernel designs usually require detailed domain-specific knowledge, and may not generalize well due to suboptimal choices of non-linearity. Additionally, kernel SVMs usually suffer from speed and memory issues when faced with large-scale data, although techniques such as non-convex optimization [6] exist for speed-ups.

Our findings suggest that the proposed model, while keeping the simplicity and efficiency of training a linear SVM, can exploit non-linear dependencies with the proposed deep architecture, as suggested by the results on two well known vision and speech datasets. In addition, our model performs better than other non-linear models under small training set sizes (i.e. it exhibits better generalization gap), which is a desirable property inherited from the linear model used in the architecture presented in the paper.

## 2    Previous Work

There has been a trend on object, acoustic and image classification to move the complexity from the classifier to the feature extraction step. The main focus of many state of the art systems has been to build rich feature descriptors (e.g. SIFT [18], HOG [7] or MFCC [8]), and use sophisticated non-linear classifiers, usually based on kernel functions and SVM or mixture models. Thus, the complexity of the overall system (feature extractor followed by the non-linear classifier) is shared in the two blocks. Vector Quantization [12], and Sparse Coding [21, 24, 26] have theoretically and empirically been shown to work well with linear classifiers. In [4], the authors note that the choice of codebook does not seem to impact performance significantly, and encoding via an inner product plus a non-linearity can effectively replace sparse coding, making testing significantly simpler and faster.

A disturbing issue with sparse coding + linear classification is that with a limited codebook size, linear separability might be an overly strong statement, undermining the use of a single linear classifier. This has been empirically verified: as we increase the codebook size, the performance keeps improving [4], indicating that such representations may not be able to fully exploit the complexity

of the data [2]. In fact, recent success on PASCAL VOC could partially be attributed to a huge codebook [25]. While this is theoretically valid, the practical advantage of linear models diminishes quickly, as the computation cost of feature generation, as well as training a high-dimensional classifier (despite linear), can make it as expensive as classical non-linear classifiers.

Despite this trend to rely on linear classifiers and overcomplete feature representations, sparse coding is still a flat model, and efforts have been made to add flexibility to the features. In particular, Deep Coding Networks [17] proposed an extension where a higher order Taylor approximation of the non-linear classification function is used, which shows improvements over coding that uses one layer. Our approach can be seen as an extension to sparse coding used in a stacked architecture.

Stacking is a general philosophy that promotes generalization in learning complex functions and that improves classification performance. The method presented in this paper is a new stacking technique that has close connections to several stacking methods developed in the literature, which are briefly surveyed in this section. In [23], the concept of stacking was proposed where simple modules of functions or classifiers are "stacked" on top of each other in order to learn complex functions or classifiers. Since then, various ways of implementing stacking operations have been developed, and they can be divided into two general categories. In the first category, stacking is performed in a layer-by-layer fashion and typically involves no supervised information. This gives rise to multiple layers in unsupervised feature learning, as exemplified in Deep Belief Networks [14, 13, 9], layered Convolutional Neural Networks [15], Deep Auto-encoder [14, 9], etc. Applications of such stacking methods includes object recognition [15, 26, 4], speech recognition [20], etc.

In the second category of techniques, stacking is carried out using supervised information. The modules of the stacking architectures are typically simple classifiers. The new features for the stacked classifier at a higher level of the hierarchy come from concatenation of the classifier output of lower modules and the raw input features. Cohen and de Carvalho [5] developed a stacking architecture where the simple module is a Conditional Random Field. Another successful stacking architecture reported in [10, 11] uses supervised information for stacking where the basic module is a simplified form of multilayer perceptron where the output units are linear and the hidden units are sigmoidal nonlinear. The linearity in the output units permits highly efficient, closed-form estimation (results of convex optimization) for the output network weights given the hidden units' outputs. Stacked context has also been used in [3], where a set of classifier scores are stacked to produce a more reliable detection. Our proposed method will build a stacked architecture where each layer is an SVM, which has proven to be a very successful classifier for computer vision applications.

## 3   The Random Recursive SVM

In this section we formally introduce the Random Recursive SVM model, and discuss the motivation and justification behind it. Specifically, we consider a training set that contains $N$ pairs of tuples $(\mathbf{d}^{(i)}, y^{(i)})$, where $\mathbf{d}^{(i)} \in \mathbb{R}^D$ is the feature vector, and $y^{(i)} \in \{1, \dots, C\}$ is the class label corresponding to the $i$-th sample.

As depicted in Figure 2(b), the model is built by multiple layers of blocks, which we call Random SVMs, that each learns a linear SVM classifier and transforms the data based on a random projection of previous layers SVM outputs. The linear SVM classifiers are learned in a one-vs-all fashion. For convenience, let $\boldsymbol{\theta} \in \mathbb{R}^{D \times C}$ be the classification matrix by stacking each parameter vector column-wise, so that $\mathbf{o}^{(i)} = \boldsymbol{\theta}^T \mathbf{d}^{(i)}$ is the vector of scores for each class corresponding to the sample $\mathbf{d}^{(i)}$, and $\hat{y}^{(i)} = \arg\max_c \boldsymbol{\theta}_c^T \mathbf{d}^{(i)}$ is the prediction for the $i$-th sample if we want to make final predictions. From this point onward, we drop the index $\cdot^{(i)}$ for the $i$-th sample for notational convenience.

### 3.1   Recursive Transform of Input Features

Figure 2(b) visualizes one typical layer in the pipeline of our algorithm. Each layer takes the output of the previous layer, (starting from $\mathbf{x}_1 = \mathbf{d}$ for the first layer as our initial input), and feeds it to a standard linear SVM that gives the output $\mathbf{o}_1$. In general, $\mathbf{o}_1$ would not be a perfect prediction, but would be better than a random guess. We then use a random projection matrix $\mathbf{W}_{2,1} \in \mathbb{R}^{D \times C}$ whose elements are sampled from $N(0, 1)$ to project the output $\mathbf{o}_1$ into the original feature space,

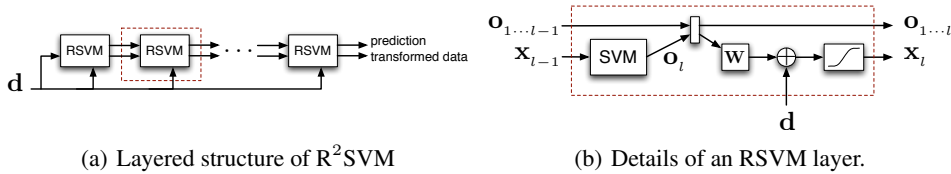

(a) Layered structure of R$^2$SVM        (b) Details of an RSVM layer.

Figure 2: The pipeline of the proposed Random Recursive SVM model. (a) The model is built with layers of Random SVM blocks, which are based on simple linear SVMs. Speech and image signals are provided as input to the first level. (b) For each random SVM layer, we train a linear SVM using the transformed data manifold by combining the original features and random projections of previous layers' predictions.

in order to use this noisy prediction to modify the original features. Mathematically, the additively modified feature space after applying the linear SVM to obtain $\mathbf{o}_1$ is:

$$\mathbf{x}_2 = \sigma(\mathbf{d} + \beta \mathbf{W}_{2,1} \mathbf{o}_1),$$

where $\beta$ is a weight parameter that controls the degree with which we move the original data sample $\mathbf{x}_1$, and $\sigma(\cdot)$ is the sigmoid function, which introduces non-linearity in a similar way as in the multilayer perceptron models, and prevents the recursive structure to degenerate to a trivial linear model. In addition, such non-linearity, akin to neural networks, has desirable properties in terms of Gaussian complexity and generalization bounds [1].

Intuitively, the random projection aims to push data from different classes towards different directions, so that the resulting features are more likely to be linearly separable. The sigmoid function controls the scale of the resulting features, and at the same time prevents the random projection to be "too confident" on some data points, as the prediction of the lower-layer is still imperfect. An important note is that, when the dimension of the feature space $D$ is relatively large, then the column vectors of $W_l$ are much likely to be approximately orthogonal, known as the quasi-orthogonality property of high-dimensional spaces [16]. At the same time, the column vectors correspond to the per class bias applied to the original sample $\mathbf{d}$ if the output was close to ideal (i.e. $\mathbf{o}_l = \mathbf{e}_c$, where $\mathbf{e}_c$ is the one-hot encoding representing class $c$), so the fact that they are approximately orthogonal means that (with high probability) they are pushing the per-class manifolds apart.

The training of the R$^2$SVM is then carried out in a purely feed-forward way. Specifically, we train a linear SVM for the $l$-th layer, and then compute the input of the next layer as the addition of the original feature space and the random projection of previous layers' outputs, which is then passed through a simple sigmoid function:

$$\mathbf{o}_l = \boldsymbol{\theta}_l^T \mathbf{x}_l$$
$$\mathbf{x}_{l+1} = \sigma(\mathbf{d} + \beta \mathbf{W}_{l+1}[\mathbf{o}_1^T, \mathbf{o}_2^T, \cdots, \mathbf{o}_l^T]^T)$$

where $\boldsymbol{\theta}_l$ are the linear SVM parameters trained with $\mathbf{x}_l$, and $\mathbf{W}_{l+1}$ is the concatenation of $l$ random projection matrices $[\mathbf{W}_{l+1,1}, \mathbf{W}_{l+1,2}, \cdots, \mathbf{W}_{l+1,l}]$, one for each previous layer, each being a random matrix sampled from $N(0,1)$.

Following [10], for each layer we use the outputs from *all* lower modules, instead of only the immediately lower module. A chief difference of our proposed method from previous approaches is that, instead of concatenating predictions with the raw input data to form the new expanded input data, we use the predictions to modify the features in the original space with a non-linear transformation. As will be shown in the next section, experimental results demonstrate that this approach is superior than simple concatenation in terms of classification performance.

### 3.2   On the Randomness in R$^2$SVM

The motivation behind our method is that projections of previous predictions help to move apart the manifolds that belong to each class in a recursive fashion, in order to achieve better linear separability (Figure 1 shows a vision example separating different image patches).

Specifically, consider that we have a two class problem which is non-linearly separable. The following Lemma illustrates the fact that, if we are given an oracle prediction of the labels, it is possible to

add an offset to each class to "pull" the manifolds apart with this new architecture, and to guarantee an improvement on the training set if we assume perfect labels.

**Lemma 3.1** *Let $\mathcal{T}$ be a set of $N$ tuples $(\mathbf{d}^{(i)}, y^{(i)})$, where $\mathbf{d}^{(i)} \in \mathbb{R}^D$ is the feature vector, and $y^{(i)} \in \{1, \ldots, C\}$ is the class label corresponding to the $i$-th sample. Let $\boldsymbol{\theta} \in \mathbb{R}^{D \times C}$ be the corresponding linear SVM solution with objective function value $f_{\mathcal{T}, \boldsymbol{\theta}}$. Then, there exist $\mathbf{w}_i \in \mathbb{R}^D$ for $i = \{1, \ldots, C\}$ such that the translated set $\mathcal{T}'$ defined as $(\mathbf{d}^{(i)} + \mathbf{w}_{y^{(i)}}, y^{(i)})$ has a linear SVM solution $\boldsymbol{\theta}'$ which achieves a better optimum $f_{\mathcal{T}', \boldsymbol{\theta}'} < f_{\mathcal{T}, \boldsymbol{\theta}}$.*

**Proof** Let $\boldsymbol{\theta}_i$ be the $i$-th column of $\boldsymbol{\theta}$ (which corresponds to the one vs all classifier for class $i$). Define $\mathbf{w}_i = \frac{\boldsymbol{\theta}_i}{||\boldsymbol{\theta}_i||_2^2}$. Then we have

$$\max(0, 1 - \boldsymbol{\theta}_{y^{(i)}}^T(\mathbf{d}^{(i)} + \mathbf{w}_{y^{(i)}})) = \max(0, 1 - (\boldsymbol{\theta}_{y^{(i)}}^T \mathbf{d}^{(i)} + 1)) \leq \max(0, 1 - (\boldsymbol{\theta}_{y^{(i)}}^T \mathbf{d}^{(i)})),$$

which leads to $f_{\mathcal{T}', \boldsymbol{\theta}} \leq f_{\mathcal{T}, \boldsymbol{\theta}}$. Since $\boldsymbol{\theta}'$ is defined to be the optimum for the set $\mathcal{T}'$, $f_{\mathcal{T}', \boldsymbol{\theta}'} \leq f_{\mathcal{T}', \boldsymbol{\theta}}$, which concludes the proof. ∎

Lemma 3.1 would work for any monotonically decreasing loss function (in particular, for the hinge loss of SVM), and motivates our search for a transform of the original features to achieve linear separability, under the guidance of SVM predictions. Note that we would achieve perfect classification under the assumption that we have oracle labels, while we only have noisy predictions for each class $\hat{y}^{(i)}$ during testing time. Under such noisy predictions, a deterministic choice of $\mathbf{w}_i$, especially linear combinations of the data as in the proof for Lemma 3.1, suffers from over-confidence in the labels and may add little benefit to the learned linear SVMs.

A first choice to avoid degenerated results is to take random weights. This enables us to use label-relevant information in the predictions, while at the same time de-correlate it with the original input $\mathbf{d}$. Surprisingly, as shown in Figure 4(a), randomness achieves a significant performance gain in contrast to the "optimal" direction given by Lemma 3.1 (which degenerates due to imperfect predictions), or alternative stacking strategies such as concatenation as in [10]. We also note that beyond sampling projection matrices from a zero-mean Gaussian distribution, a biased sampling that favors directions near the "optimal" direction may also work, but the degree of bias would be empirically difficult to determine and may be data-dependent. In general, we aim to avoid supervision in the projection parameters, as trying to optimize the weights jointly would defeat the purpose of having a computationally efficient method, and would, perhaps, increase training accuracy at the expense of over-fitting. The risk of over-fitting is also lower in this way, as we do not increase the dimensionality of the input space, and we do not learn the matrices $\mathbf{W}_l$, which means we pass a weak signal from layer to layer. Also, training Random Recursive SVM is carried out in a feed-forward way, where each step involves a convex optimization problem that can be efficiently solved.

### 3.3 Synthetic examples

To visually show the effectiveness of our approach in learning non-linear SVM classifiers without kernels, we apply our algorithm to two synthetic examples, neither of which can be linearly separated. The first example contains two classes distributed in a two-moon shaped way, and the second example contains data distributed as two more complex spirals. Figure 3 visualizes the classification hyperplane at different stages of our algorithm. The first layer of our approach is identical to the linear SVM, which is not able to separate the data well. However, when classifiers are recursively stacked in our approach, the classification hyperplane is able to adapt to the nonlinear characteristics of the two classes.

## 4   Experiments

In this section we empirically evaluate our method, and support our claims: (1) for low-dimensional features, linear SVMs suffer from their limited representation power, while $R^2$SVMs significantly improve performance; (2) for high-dimensional features, and especially when faced with limited amount of training data, $R^2$SVMs exhibit better generalization power than conventional kernelized non-linear SVMs; and (3) the random, feed-forward learning scheme is able to achieve state-of-the-art performance, without complex fine-tuning.

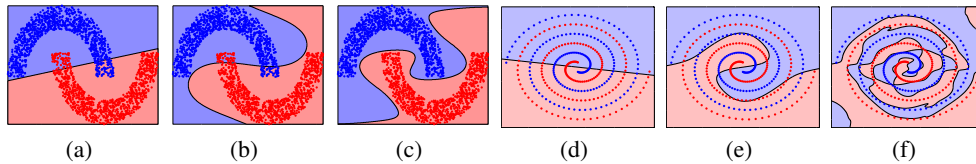

(a)　　　(b)　　　(c)　　　(d)　　　(e)　　　(f)

Figure 3: Classification hyperplane from different stages of our algorithm: first layer, second layer, and final layer outputs. (a)-(c) show the two-moon data and (d)-(f) show the spiral data.

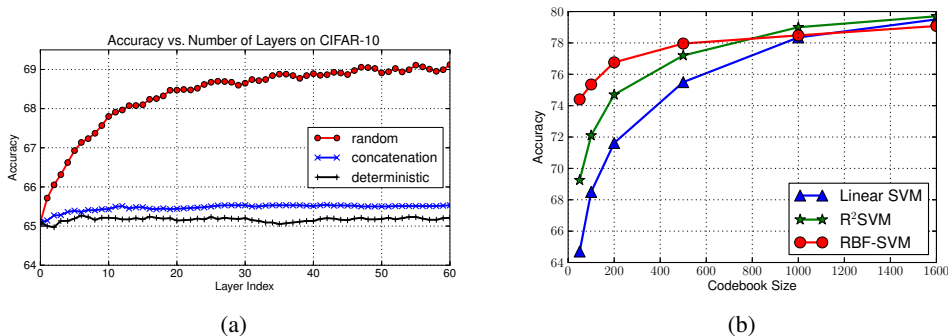

(a)　　　　　　　　　　(b)

Figure 4: Results on CIFAR-10. (a) Accuracy versus number of layers on CIFAR-10 for Random Recursive SVM with all the training data and 50 codebook size, for a baseline where the output of a classifier is concatenated with the input feature space, and for a deterministic version of recursive SVM where the projections are as in the proof of Lemma 3.1. (b) Accuracy versus codebook size on CIFAR-10 for linear SVM, RBF SVM, and our proposed method.

We describe the experimental results on two well known classification benchmarks: CIFAR-10 and TIMIT. The CIFAR-10 dataset contains large amount of training/testing data focusing on object classification. TIMIT is a speech database that contains two orders of magnitude more training samples than the other datasets, and the largest output label space.

Recall that our method relies on two parameters: $\beta$, which is the factor that controls how much to shift the original feature space, and $C$, the regularization parameter of the linear SVM trained at each layer. $\beta$ is set to $\frac{1}{10}$ for all the experiments, which was experimentally found to work well for one of the CIFAR-10 configurations. $C$ controls the regularization of each layer, and is an important parameter – setting it too high will yield overfitting as the number of layers is increased. As a result, we learned this parameter via cross validation for each configuration, which is the usual practice of other approaches. Lastly, for each layer, we sample a new random matrix $\mathbf{W}_l$. As a result, even if the training and testing sets are fixed, randomness still exists in our algorithm. Although one may expect the performance to fluctuate from run to run, in practice we never observe a standard deviation larger than $0.25$ (and typically less than $0.1$) for the classification accuracy, over multiple runs of each experiment.

**CIFAR-10**

The CIFAR-10 dataset contains 10 object classes with a fair amount of training examples per class (5000), with images of small size (32x32 pixels). For this dataset, we follow the standard pipeline defined in [4]: dense 6x6 local patches with ZCA whitening are extracted with stride 1, and thresholding coding with $\alpha = 0.25$ is adopted for encoding. The codebook is trained with OMP-1. The features are then average-pooled on a $2 \times 2$ grid to form the global image representation. We tested three classifiers: linear SVM, RBF kernel based SVM, and the Random Recursive SVM model as introduced in Section 3.

As have been shown in Figure 4(b), the performance is almost monotonically increasing as we stack more layers in R$^2$SVM. Also, stacks of SVMs by concatenation of output and input feature space does not yield much gain above 1 layer (which is a linear SVM), and neither does a deterministic

Table 1: Results on CIFAR-10, with different codebook sizes (hence feature dimensions).

| Method | Tr. Size | Code. Size | Acc. |
|---|---|---|---|
| Linear SVM | All | 50 | 64.7% |
| RBF SVM | All | 50 | 74.4% |
| $R^2$SVM | All | 50 | 69.3% |
| DCN | All | 50 | 67.2% |
| Linear SVM | All | 1600 | 79.5% |
| RBF SVM | All | 1600 | 79.0% |
| $R^2$SVM | All | 1600 | 79.7% |
| DCN | All | 1600 | 78.1% |

Table 2: Results on CIFAR-10, with 25 training data per class.

| Method | Tr. Size | Code. Size | Acc. |
|---|---|---|---|
| Linear SVM | 25/class | 50 | 41.3% |
| RBF SVM | 25/class | 50 | 42.2% |
| $R^2$SVM | 25/class | 50 | 42.8% |
| DCN | 25/class | 50 | 40.7% |
| Linear SVM | 25/class | 1600 | 44.1% |
| RBF SVM | 25/class | 1600 | 41.6% |
| $R^2$SVM | 25/class | 1600 | 45.1% |
| DCN | 25/class | 1600 | 42.7% |

version of recursive SVM where a projection matrix as in the proof for Lemma 3.1 is used. For the $R^2$SVM, in most cases the performance asymptotically converges within 30 layers. Note that training each layer involves training a linear SVM, so the computational complexity is simply linear to the depth of our model. In contrast to this, the difficulty of training deep learning models based on many hidden layers may be significantly harder, partially due to the lack of supervised information for its hidden layers.

Figure 4(b) shows the effect that the feature dimensionality (controlled by the codebook size of OMP-1) has on the performance of the linear and non-linear classifiers, and Table 1 provides representative numerical results. In particular, when the codebook size is low, the assumption that we can approximate the non-linear function $f$ as a globally linear classifier fails, and in those cases the $R^2$SVM and RBF SVM clearly outperform the linear SVM. Moreover, as the codebook size grows, non-linear classifiers, represented by RBF SVM in our experiments, suffer from the curse of dimensionality partially due to the large dimensionality of the over-complete feature representation. In fact, as the dimensionality of the over-complete representation becomes too large, RBF SVM starts performing worse than linear SVM. For linear SVM, increasing the codebook size makes it perform better with respect to non-linear classifiers, but additional gains can still be consistently obtained by the Random Recursive SVM method. Also note how our model outperforms DCN, another stacking architecture proposed in [10].

Similar to the change of codebook sizes, it is interesting to experiment with the number of training examples per class. In the case where we use fewer training examples per class, little gain is obtained by classical RBF SVMs, and performance even drops when the feature dimension is too high (Table 2), while our Random Recursive SVM remains competitive and does not overfit more than any baseline. This again suggests that our proposed method may generalize better than RBF, which is a desirable property when the number of training examples is small with respect to the dimensionality of the feature space, which are cases of interest to many computer vision applications.

In general, our method is able to combine the advantages of both linear and nonlinear SVM: it has higher representation power than linear SVM, providing consistent performance gains, and at the same time has a better robustness against overfitting. It is also worth pointing out again that $R^2$SVM is highly efficient, since each layer is a simple linear SVM that can be carried out by simple matrix multiplication. On the other hand, non-linear SVMs like RBF SVM may take much longer to run especially for large-scale data, when special care has to be taken [6].

**TIMIT**

Finally, we report our experiments using the popular speech database TIMIT. The speech data is analyzed using a 25-ms Hamming window with a 10-ms fixed frame rate. We represent the speech using first- to 12th-order Mel frequency cepstral coefficients (MFCCs) and energy, along with their first and second temporal derivatives. The training set consists of 462 speakers, with a total number of frames in the training data of size 1.1 million, making classical kernel SVMs virtually impossible to train. The development set contains 50 speakers, with a total of 120K frames, and is used for cross validation. Results are reported using the standard 24-speaker core test set consisting of 192 sentences with 7333 phone tokens and 57920 frames.

The data is normalized to have zero mean and unit variance. All experiments used a context window of 11 frames. This gives a total of $39 \times 11 = 429$ elements in each feature vector. We used 183

Table 3: Performance comparison on TIMIT.

| Method | Phone state accuracy |
|---|---|
| Linear SVM | 50.1% (2000 codes) 53.5% (8000 codes) |
| $R^2$SVM | 53.5% (2000 codes) 55.1% (8000 codes) |
| DCN, learned per-layer | 48.5% |
| DCN, jointly fine-tuned | 54.3% |

target class labels (i.e., three states for each of the 61 phones), which are typically called "phone states", with a one-hot encoding.

The pipeline adopted is otherwise unchanged from the previous dataset. However, we did not apply pooling, and instead coded the whole $429$ dimensional vector with a dictionary with $2000$ and $8000$ elements found with OMP-1, with the same parameter $\alpha$ as in the vision tasks. The competitive results with a framework known in vision adapted to speech [22], as shown in Table 3, are interesting on their own right, as the optimization framework for linear SVM is well understood, and the dictionary learning and encoding step are almost trivial and scale well with the amounts of data available in typical speech tasks. On the other hand, our $R^2$SVM boosts performance quite significantly, similar to what we observed on other datasets.

In Table 3 we also report recent work on this dataset [10], which uses multi-layer perceptron with a hidden layer and linear output, and stacks each block on top of each other. In their experiments, the representation used from the speech signal is not sparse, and uses instead Restricted Boltzman Machine, which is more time consuming to learn. In addition, only when jointly optimizing the network weights (fine tuning), which requires solving a non-convex problem, the accuracy achieves state-of-the-art performance of $54.3\%$. Our method does not include this step, which could be added as future work; we thus think the fairest comparison of our result is to the per-layer DCN performance.

In all the experiments above we have observed two advantages of $R^2$SVM. First, it provides a consistent improvement over linear SVM. Second, it can offer a better generalization ability over non-linear SVMs, especially when the ratio of dimensionality to the number of training data is large. These advantages, combined with the fact that $R^2$SVM is efficient in both training and testing, suggests that it could be adopted as an improvement over the existing classification pipeline in general.

We also note that in the current work we have not employed techniques of fine tuning similar to the one employed in the architecture of [10]. Fine tuning of the latter architecture has accounted for between 10% to 20% error reduction, and reduces the need for having large depth in order to achieve a fixed level of recognition accuracy. Development of fine-tuning is expected to improve recognition accuracy further, and is in the interest of future research. However, even without fine tuning, the recognition accuracy is still shown to consistently improve until convergence, showing the robustness of the proposed method.

## 5 Conclusions and Future Work

In this paper, we investigated low level vision and audio representations. We combined the simplicity of linear SVMs with the power derived from deep architectures, and proposed a new stacking technique for building a better classifier, using linear SVM as the base building blocks and emplying a random non-linear projection to add flexibility to the model. Our work is partially motivated by the recent trend of using coding techniques as feature representation with relatively large dictionaries. The chief advantage of our method lies in the fact that it learns non-linear classifiers without the need of kernel design, while keeping the efficiency of linear SVMs. Experimental results on vision and speech datasets showed that the method provides consistent improvement over linear baselines, even with no learning of the model parameters. The convexity of our model could lead to better theoretical analysis of such deep structures in terms of generalization gap, adds interesting opportunities for learning using large computer clusters, and would potentially help understanding the nature of other deep learning approaches, which is the main interest of future research.

# References

[1] P L Bartlett and S Mendelson. Rademacher and gaussian complexities: Risk bounds and structural results. *The Journal of Machine Learning Research*, 3:463–482, 2003.

[2] O Boiman, E Shechtman, and M Irani. In defense of nearest-neighbor based image classification. In *CVPR*, 2008.

[3] L Bourdev, S Maji, T Brox, and J Malik. Detecting people using mutually consistent poselet activations. In *ECCV*, 2010.

[4] A Coates and A Ng. The importance of encoding versus training with sparse coding and vector quantization. In *ICML*, 2011.

[5] W Cohen and V R de Carvalho. Stacked sequential learning. In *IJCAI*, 2005.

[6] R Collobert, F Sinz, J Weston, and L Bottou. Trading convexity for scalability. In *ICML*, 2006.

[7] N Dalal. Histograms of oriented gradients for human detection. In *CVPR*, 2005.

[8] S Davis and P Mermelstein. Comparison of parametric representations for monosyllabic word recognition in continuously spoken sentences. *Acoustics, Speech and Signal Processing, IEEE Transactions on*, 28(4):357–366, 1980.

[9] L Deng, M L Seltzer, D Yu, A Acero, A Mohamed, and G Hinton. Binary coding of speech spectrograms using a deep auto-encoder. In *Interspeech*, 2010.

[10] L Deng and D Yu. Deep convex network: A scalable architecture for deep learning. In *Interspeech*, 2011.

[11] L Deng, D Yu, and J Platt. Scalable stacking and learning for building deep architectures. In *ICASSP*, 2012.

[12] L Fei-Fei and P Perona. A bayesian hierarchical model for learning natural scene categories. In *CVPR*, 2005.

[13] G Hinton, L Deng, D Yu, G Dahl, A Mohamed, N Jaitly, A Senior, V Vanhoucke, P Nguyen, T Sainath, and B Kingsbury. Deep Neural Networks for Acoustic Modeling in Speech Recognition. *IEEE Signal Processing Magazine*, 28:82–97, 2012.

[14] G Hinton and R Salakhutdinov. Reducing the dimensionality of data with neural networks. *Science*, 313(5786):504, 2006.

[15] K Jarrett, K Kavukcuoglu, M A Ranzato, and Y LeCun. What is the best multi-stage architecture for object recognition? In *ICCV*, 2009.

[16] T Kohonen. *Self-Organizing Maps*. Springer-Verlag, 2001.

[17] Y Lin, T Zhang, S Zhu, and K Yu. Deep coding network. In *NIPS*, 2010.

[18] D Lowe. Distinctive image features from scale-invariant keypoints. *IJCV*, 2004.

[19] S Maji, AC Berg, and J Malik. Classification using intersection kernel support vector machines is efficient. In *Computer Vision and Pattern Recognition, 2008. CVPR 2008. IEEE Conference on*, pages 1–8. Ieee, 2008.

[20] A Mohamed, D Yu, and L Deng. Investigation of full-sequence training of deep belief networks for speech recognition. In *Interspeech*, 2010.

[21] B Olshausen and D J Field. Sparse coding with an overcomplete basis set: a strategy employed by V1? *Vision research*, 37(23):3311–3325, 1997.

[22] O Vinyals and L Deng. Are Sparse Representations Rich Enough for Acoustic Modeling? In *Interspeech*, 2012.

[23] D H Wolpert. Stacked generalization. *Neural networks*, 5(2):241–259, 1992.

[24] J Yang, K Yu, and Y Gong. Linear spatial pyramid matching using sparse coding for image classification. In *CVPR*, 2009.

[25] J Yang, K Yu, and T Huang. Efficient highly over-complete sparse coding using a mixture model. In *ECCV*, 2010.

[26] K Yu and T Zhang. Improved Local Coordinate Coding using Local Tangents. In *ICML*, 2010.

